# Stationarity of Synaptic Coupling Strength Between Neurons with Nonstationary Discharge Properties

**Mark R. Sydorenko and Eric D. Young**
Dept. of Biomedical Engineering & Center for Hearing Sciences
The Johns Hopkins School of Medicine
720 Rutland Avenue
Baltimore, Maryland 21205

## Abstract

Based on a general non-stationary point process model, we computed estimates of the synaptic coupling strength (efficacy) as a function of time after stimulus onset between an inhibitory interneuron and its target postsynaptic cell in the feline dorsal cochlear nucleus. The data consist of spike trains from pairs of neurons responding to brief tone bursts recorded *in vivo*. Our results suggest that the synaptic efficacy is non-stationary. Further, synaptic efficacy is shown to be inversely and approximately linearly related to average presynaptic spike rate. A second-order analysis suggests that the latter result is not due to non-linear interactions. Synaptic efficacy is less strongly correlated with postsynaptic rate and the correlation is not consistent across neural pairs.

## 1 INTRODUCTION

The aim of this study was to investigate the dynamic properties of the inhibitory effect of type II neurons on type IV neurons in the cat dorsal cochlear nucleus (DCN). Type IV cells are the principal (output) cells of the DCN and type II cells are inhibitory interneurons (Voigt & Young 1990). In particular, we examined the stationarity of the efficacy of inhibition of neural activity in a type IV neuron by individual action potentials (APs) in a type II neuron. Synaptic efficacy, or *effectiveness*, is defined as the average number of postsynaptic (type IV) APs eliminated per presynaptic (type II) AP .

This study was motivated by the observation that post-stimulus time histograms of type IV neurons often show gradual recovery ("buildup") from inhibition (Rhode et al. 1983; Young & Brownell 1976) which could arise through a weakening of inhibitory input over time.

Correlograms of pairs of DCN units using long duration stimuli are reported to display inhibitory features (Voigt & Young 1980; Voigt & Young 1990) whereas correlograms using short stimuli are reported to show excitatory features (Gochin et al. 1989). This difference might result from nonstationarity of synaptic coupling. Finally, pharmacological results (Caspary et al. 1984) and current source-density analysis of DCN responses to electrical stimulation (Manis & Brownell 1983) suggest that this synapse may fatigue with activity.

Synaptic efficacy was investigated by analyzing the statistical relationship of spike trains recorded simultaneously from pairs of neurons *in vivo* . We adopt a first order (linear) non-stationary point process model that does not impose *a priori* restrictions on the presynaptic process's distribution. Using this model, estimators of the postsynaptic impulse response to a presynaptic spike were derived using martingale theory and a method of moments approach. To study stationarity of synaptic efficacy, independent estimates of the impulse response were derived over a series of brief time windows spanning the stimulus duration. Average pre- and postsynaptic rate were computed for each window, as well. In this report, we summarize the results of analyzing the dependence of synaptic efficacy (derived from the impulse response estimates) on post-stimulus onset time, presynaptic average rate, postsynaptic average rate, and presynaptic interspike interval.

## 2    METHODS

### 2.1    DATA COLLECTION

Data were collected from unanesthetized cats that had been decerebrated at the level of the superior colliculus. We used a posterior approach to expose the DCN that did not require aspiration of brain tissue nor disruption of the local blood supply. Recordings were made using two platinum-iridium electrodes.

The electrodes were advanced independently until a type II unit was isolated on one electrode and a type IV unit was isolated on the other electrode. Only pairs of units with best frequencies (BFs) within 20% were studied. The data consist of responses of the two units to 500-4000 repetitions of a 100-1500 millisecond tone. The frequency of the tone was at the type II BF and the tone level was high enough to elicit activity in the type II unit for the duration of the presentation, but low enough not to inhibit the activity of the type IV unit (usually 5-10 dB above the type II threshold). Driven discharge rates of the two units ranged from 15 to 350 spikes per second. A silent recovery period at least four times longer than the tone burst duration followed each stimulus presentation.

### 2.3    DATA ANALYSIS

The stimulus duration is divided into 3 to 9 overlapping or non-overlapping time windows ('a' thru 'k' in figure 1). A separate impulse response estimate, presynaptic rate, and postsynaptic rate computation is made using only those type II and type IV spikes that fall within each window. The effectiveness of synaptic coupling during each window is calculated from the area bounded by the impulse response feature and the abscissa (shaded area in figure 1). The effectiveness measure has units of number of spikes.

The synaptic impulse response is estimated using a non-stationary method of moments algorithm. The estimation algorithm is based on the model depicted in figure 2. The thick gray line encircles elements belonging to the postsynaptic (type IV) cell. The neural network surrounding the postsynaptic cell is modelled as a J-dimensional multivariate counting process. Each element of the J-dimensional counting process is an input to the postsynaptic

cell. One of these input elements is the presynaptic (type II) cell under observation. The input processes modulate the postsynaptic cell's instantaneous rate function, $\lambda_j(t)$. Roughly speaking, $\lambda_j(t)$ is the conditional firing probability of neuron j given the history of the input events up to time t.

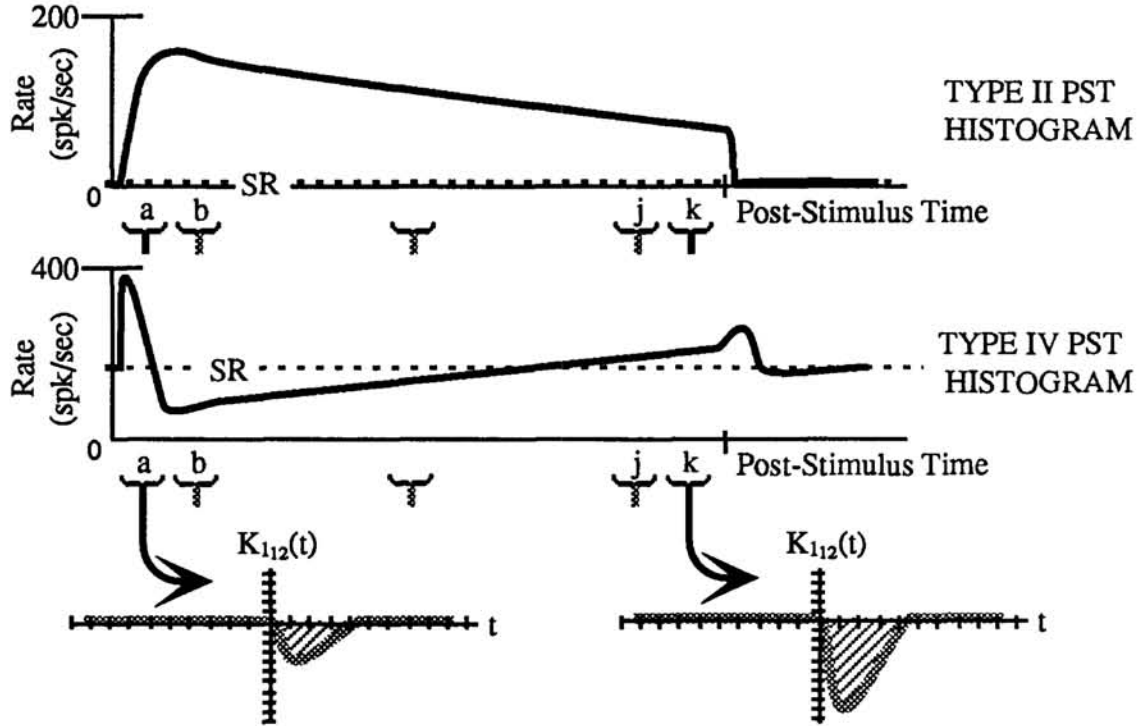

Figure 1: Analysis of Non-stationary Synaptic Coupling

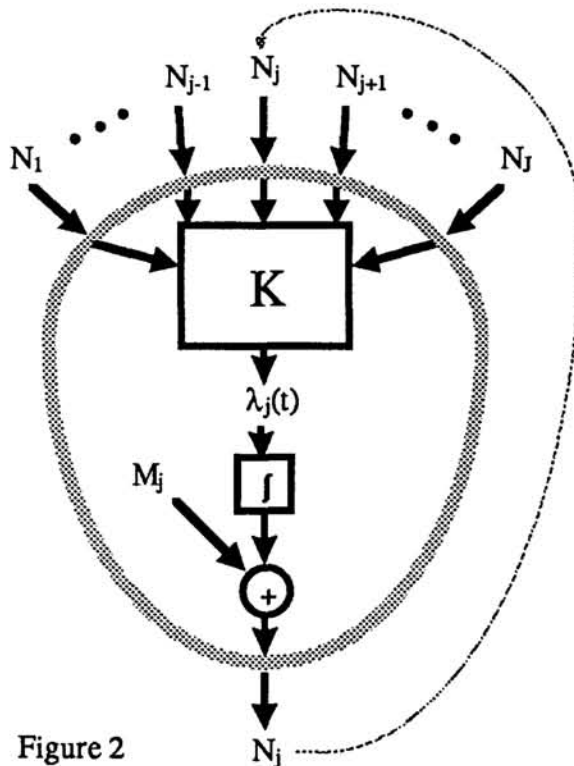

Figure 2

The transformation K describes how the input processes influence $\lambda_j(t)$. We model this transformation as a linear sum of an intrinsic rate component and the contribution of all the presynaptic processes:

$$\lambda_j(t) = K_{0j}(t) + \sum_{k=1}^{J} \int K_{1jk}(t,u)\, dN_k(u) \tag{1}$$

where $K_0$ describes the intrinsic rate and the $K_1$ describe the impulse response of the postsynaptic cell in response to an input event. The output of the postsynaptic neuron is modeled as the integral of this rate function plus a mean-zero noise process, the innovation martingale (Bremaud 1981):

$$N_j(t) = \int_{T_0}^{t} \lambda_j(u)\, du + M_j(t). \tag{2}$$

An algorithm for estimating the first order kernel, $K_1$, was derived without assuming

anything about the distribution of the presynaptic process and without assuming stationary first or second order product densities (i.e., without assuming stationary rate or stationary auto-correlation). One or more such assumptions have been made in previous method of moments based algorithms for estimating neural interactions (Chornoboy et al. 1988 describe a maximum likelihood approach that does not require these assumptions).

Since $K_1$ is assumed to be stationary during the windowed interval (figure 1) while the process product densities are non-stationary (see PSTHs in figure 1), $K_1$ is an average of separate estimates of $K_1$ computed at each point in time during the windowed interval:

$$\widehat{K}_{1_{ij}}(t^\Delta) = \frac{1}{n^\Delta} \sum_{t_i^\Delta - t_j^\Delta = t^\Delta;\, t_j^\Delta \in I} \widehat{K}_{1_{ij}}(t_i^\Delta, t_j^\Delta) \tag{3}$$

where $K_1$ inside the summation is an estimate of the impulse response of neuron i at time $t_i^\Delta$ to a spike from neuron j at time $t_j^\Delta$ (times are relative to stimulus onset); the digitization bin width $\Delta$ (= 0.3 msec in our case) determines the location of the discrete time points as well as the number of separate kernel estimates, $n^\Delta$, within the windowed interval, I. The time dependent kernel, $K_1(\cdot,\cdot)$, is computed by deconvolving the effects of the presynaptic process distribution, described by $r_{ii}$ below, from the estimate of the cross-cumulant density, $q_{ij}$:

$$\widehat{K}_{1_{ij}}(t_i^\Delta, t_j^\Delta) = \sum_{v^\Delta} \widehat{q}_{ij}(v^\Delta, t_j^\Delta)\, \widehat{r}_{jj}^{-1}(t_i^\Delta - v^\Delta, t_j^\Delta)\, \Delta \tag{4}$$

where:

$$\widehat{q}_{ij}(u^\Delta, v^\Delta) = \widehat{p}_{ij}(u^\Delta, v^\Delta) - \widehat{p}_i(u^\Delta)\widehat{p}_j(v^\Delta), \tag{5}$$

$$\widehat{r}_{jj}(u^\Delta, v^\Delta) = \widehat{q}_{jj}(u^\Delta, v^\Delta) + \delta(u^\Delta - v^\Delta)\widehat{p}_j(v^\Delta), \tag{6}$$

$$\widehat{r}_{jj}^{-1}(u^\Delta, v^\Delta) = \mathcal{F}^{-1}\left[ \frac{1}{\mathcal{F}[\widehat{r}_{jj}(u^\Delta, v^\Delta)]} \right], \tag{7}$$

$$\widehat{p}_j(t_j^\Delta) = \#\left\{\text{spike in neuron j during } [t_j^\Delta - \tfrac{\Delta}{2}, t_j^\Delta + \tfrac{\Delta}{2})\right\} \Big/ \left( \#\{\text{trials}\}\, \Delta \right), \tag{8}$$

$$\widehat{p}_{ij}(t_i^\Delta, t_j^\Delta) = \frac{\#\left\{\text{spike in i during } [t_i^\Delta - \tfrac{\Delta}{2}, t_i^\Delta + \tfrac{\Delta}{2}) \text{ and spike in j during } [t_j^\Delta - \tfrac{\Delta}{2}, t_j^\Delta + \tfrac{\Delta}{2})\right\}}{\#\{\text{trials}\}\, \Delta^2}, \tag{9}$$

where $\delta(\cdot)$ is the dirac delta function; $\mathcal{F}$ and $\mathcal{F}^{-1}$ are the DFT and inverse DFT, respectively; and $\#\{\cdot\}$ is the number of members in the set described inside the braces. If the presynaptic process is Poisson distributed, expression (4) simplifies to:

$$\widehat{K}_{1_{ij}}(t_i^\Delta, t_j^\Delta) = \frac{\widehat{q}_{ij}(t_i^\Delta, t_j^\Delta)}{\widehat{p}_j(t_j^\Delta)} \tag{10}$$

Under mild (physiologically justifiable) conditions, the estimator given by (3) converges in quadratic mean and yields an asymptotically unbiased estimate of the true impulse response function (in the general, (4), and Poisson presynaptic process, (10), cases).

## 3    RESULTS

Figure 3 displays estimates of synaptic impulse response functions computed using traditional cross-correlation analysis and compares them to estimates computed using the method of moments algorithms described above. (We use the definition of cross-correlation given by Voigt & Young 1990; equivalent to the function given by dividing expression (10) by

expression (9) after averaging across all $t_j$.) Figure 3A compares estimates computed from the responses of a real type II and type IV unit during the first 15 milliseconds of stimulation (where nonstationarity is greatest). Note that the cross-correlation estimate is distorted due to the nonstationarity of the underlying processes. This distortion leads to an overestimation of the effectiveness measure (shaded area) as compared to that yielded by the method of moments algorithm below. Figure 3B compares estimates computed using a simulated data set where the presynaptic neuron had regular (non-Poisson) discharge properties. Note the characteristic ringing pattern in the cross-correlation estimate as well as the larger feature amplitude in the non-Poisson method of moments estimate.

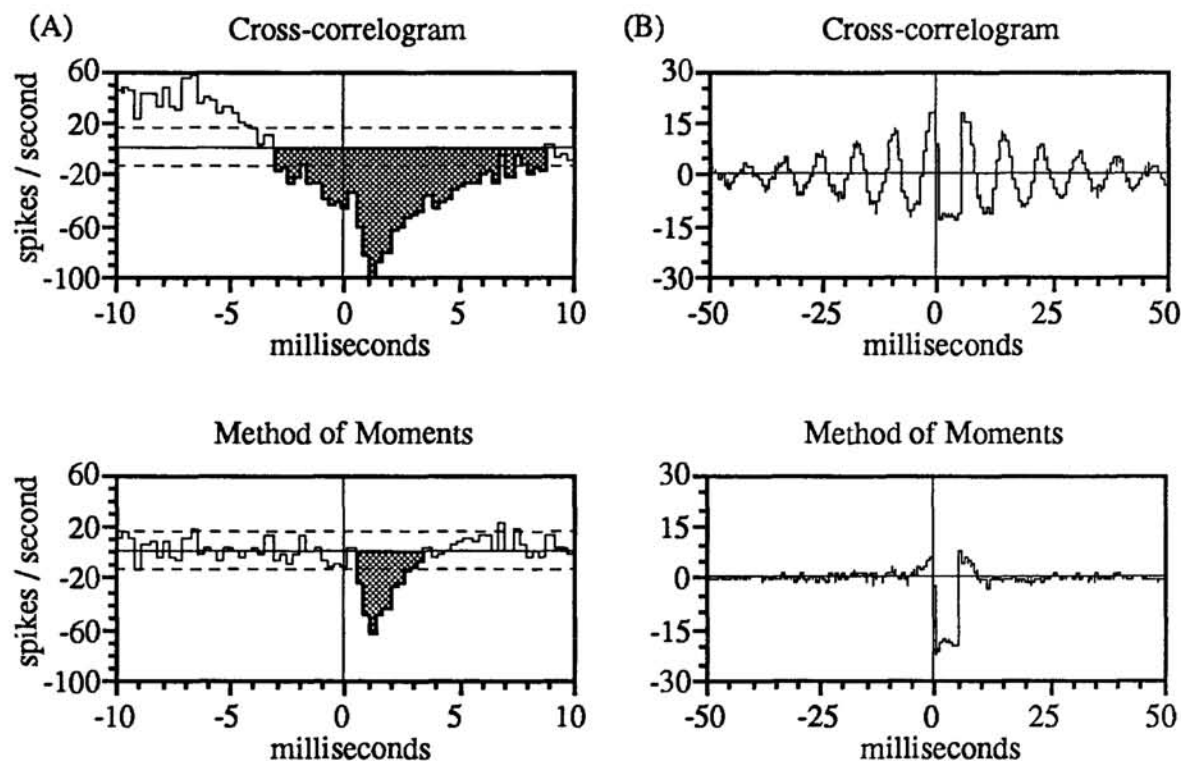

Figure 3

Results from one analysis of eight different type II / type IV pairs are shown in figure 4. For each pair, the effectiveness and the presynaptic (type II) average rate during each window are plotted and fit with a least squares line. Similar analyses were performed for effectiveness versus postsynaptic rate and for effectiveness versus post-stimulus-onset time. The number of pairs showing a positive or negative correlation of effectiveness with each parameter are tallied of table 1. The last column shows the average correlation coefficient of the lines fit to the eight sets of data. Note that: Synaptic efficacy tends to increase with time; there is no consistent relationship between synaptic efficacy and postsynaptic rate; there is a strong inverse and linear correlation between synaptic efficacy and presynaptic rate in 7 out of 8 pairs.

If the data appearing in figure 4 had been plotted as effectiveness versus average interspike interval (reciprocal of average rate) of the presynaptic neuron, the result would suggest that synaptic efficacy increases with average inter-spike interval. This result would be consistent with the interpretation that the effectiveness of an input event is suppressed by the occurrence of an input event immediately before it. The linear model initially used to analyze these data neglects the possibility of such second order effects.

Table 1:  Summary of Results

| GRAPH | NUMBER OF PAIRS WITH *POSITIVE* SLOPE | NUMBER OF PAIRS WITH *NEGATIVE* SLOPE | AVERAGE LINEAR REGRESSION CORRELATION COEFFICIENT |
|---|---|---|---|
| Effectiveness -vs- Post Stimulus Onset Time | 7 / 8 | 1 / 8 | 0.83 |
| Effectiveness -vs- Average Postsynaptic Rate | 5 / 8 | 3 / 8 | 0.72 |
| Effectiveness -vs- Average Presynaptic Rate | 1 / 8 | 7 / 8 | **0.89** |

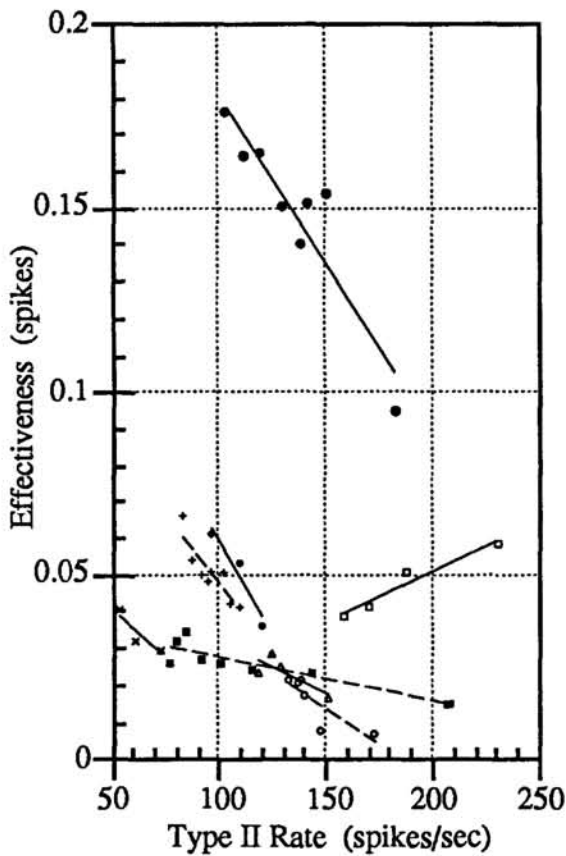

Figure 4

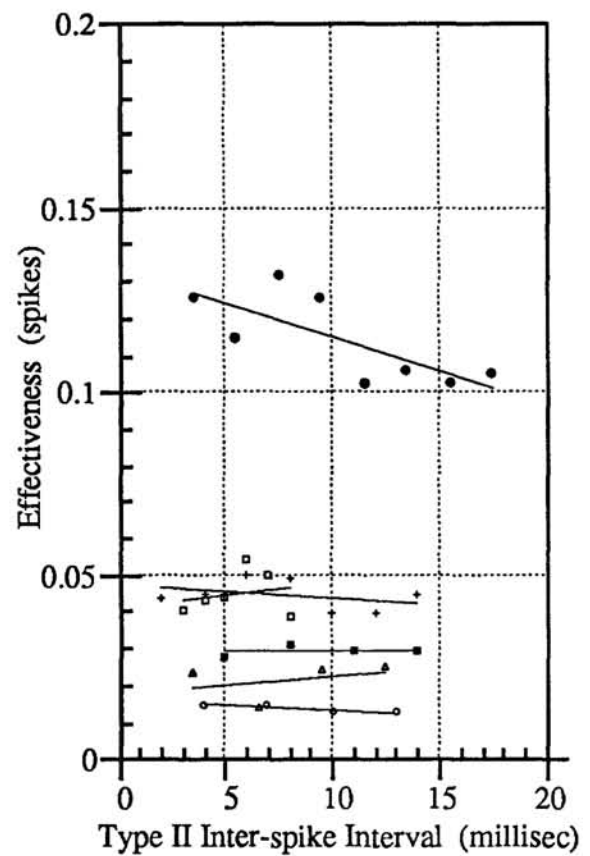

Figure 5

We used a modification of the analysis described in the methods to investigate second order effects.  Rather than window small segments of the stimulus duration as in figure 1, the entire duration was used in this analysis.  Impulse response estimates were constructed conditional

on presynaptic interspike interval. For example, the first estimate was constructed using presynaptic events occurring after a 1 ms interspike interval, the second estimate was based on events after a 2 ms interval, and so on.

The results of the second order analysis are shown in figure 5. Note that there is no systematic relationship between conditioning interspike interval and effectiveness. In fact, lines fitted to these points tend to be horizontal, suggesting that there are no significant second order effects under these experimental conditions.

Our results suggest that synaptic efficacy is inversely and roughly linearly related to average presynaptic rate. We have attempted to understand the mechanism of the observed decrease in efficacy in terms of a model that assumes stationary synaptic coupling mechanisms. The model was designed to address the following hypothesis: Could the decrease in synaptic efficacy at high input rates be due to an increase in the likelihood of driving the stochastic intensity below zero, and, hence decreasing the apparent efficacy of the input due to clipping? The answer was pursued by attempting to reproduce the data collected for the 3 best type II / type IV pairs in our data set. Real data recorded from the presynaptic unit are used as input to these models. The parameters of the models were adjusted so that the first moment of the output process had the same quantitative trajectory as that seen in the real postsynaptic unit. The simulated data were analyzed by the same algorithms used to analyze the real data. Our goal was to compare the simulated results with the real results. If the simulated data showed the same inverse relationship between presynaptic rate and synaptic efficacy as the real data, it would suggest that the phenomenon is due to non-linear clipping by the postsynaptic unit.

The simulation algorithm was based on the model described in figure 2 and equation (1) but with the following modifications:

- The experimentally determined type IV PST profile was substituted for $K_0$ (this term represents the average combined influence of all extrinsic inputs to the type IV cell plus the intrinsic spontaneous rate).
- An impulse response function estimated from the data was substituted for $K_1$ (this kernel is stationary in the simulation model).
- The convolution of the experimentally determined type II spikes with the first-order kernel was used to perturb the output cell's stochastic intensity:

$$\lambda_1(t) \;=\; \text{MAX} \left[ 0, \; p_1(t) + \sum_{dN_2(u_i)=\delta} K_{1_{12}}(t - u_i) \right]$$

where:  $dN_2(t)$ = Real type II cell spike record, and
   $p_1(t)$ = PST profile of real type IV cell.

- The output process was simulated as a non-homogeneous Poisson process with $\lambda_1(t)$ as its parameter. This process was modified by a 0.5 msec absolute dead time.
- The simulated data were analyzed in the same manner as the real data.

The dependence of synaptic efficacy on presynaptic rate in the simulated data was compared to the corresponding real data. In 1 out of the 3 cases, we observed an inverse relationship between input rate and efficacy despite the use of a stationary first order kernel in the simulation. The similarity between the real and simulated results for this one case suggests that the mechanism may be purely statistical rather than physiological (e.g., not presynaptic depletion or postsynaptic desensitization). The other 2 simulations did not yield a strong dependence of effectiveness on input rate and, hence, failed to mimic the experimental results. In these two cases, the results suggest that the mechanism is not due solely to clipping, but involves some additional, possibly physiological, mechanisms.

## 4    CONCLUSIONS

1)  The amount of inhibition imparted to type IV units by individual presynaptic type II unit action potentials (expressed as the expected number of type IV spikes eliminated per type II spike) is inversely and roughly linearly related to the average rate of the type II unit.

(2)  There is no evidence for second order synaptic effects at the type II spike rates tested. In other words, the inhibitory effect of two successive type II spikes is simply the linear sum of the inhibition imparted by each individual spike.

(3)  There is no consistent relationship between type II / type IV synaptic efficacy and postsynaptic (type IV) rate.

(4)  Simulations, in some cases, suggest that the inverse relationship between presynaptic rate and effectiveness may be reproduced using a simple statistical model of neural interaction.

(5)  We found no evidence that would explain the discrepancy between Voigt and Young's results and Gochin's results in the DCN. Gochin observed correlogram features consistent with monosynaptic excitatory connections within the DCN when short tone bursts were used as stimuli. We did not observe excitatory features between any unit pairs using short tone bursts.

**Acknowledgements**

Dr. Alan Karr assisted in developing Eqns. 1-10. E. Nelken provided helpful comments. Research supported by NIH grant DC00115.

**References**

Bremaud, P. (1981). Point Processes and Queues: Martingale Dynamics. New York, Springer-Verlag.

Caspary, D.M., Rybak, L.P.et al. (1984). "Baclofen reduces tone-evoked activity of cochlear nucleus neurons." Hear Res. 13: 113-22.

Chornoboy, E.S., Schramm, L.P.et al. (1988). "Maximum likelihood identification of neural point process systems." Biol Cybern. 59: 265-75.

Gochin, P.M., Kaltenbach, J.A.et al. (1989). "Coordinated activity of neuron pairs in anesthetized rat dorsal cochlear nucleus." Brain Res. 497: 1-11.

Manis, P.B. & Brownell, W.E. (1983). "Synaptic organization of eighth nerve afferents to cat dorsal cochlear nucleus." J Neurophysiol. 50: 1156-81.

Rhode, W.S., Smith, P.H.et al. (1983). "Physiological response properties of cells labeled intracellularly with horseradish peroxidase in cat dorsal cochlear nucleus." J Comp Neurol. 213: 426-47.

Voigt, H.F. & Young, E.D. (1980). "Evidence of inhibitory interactions between neurons in dorsal cochlear nucleus." J Neurophys. 44: 76-96.

Voigt, H.F. & Young, E.D. (1990). "Cross-correlation analysis of inhibitory interactions in the Dorsal Cochlear Nucleus." J Neurophys. 54: 1590-1610.

Young, E.D. & Brownell, W.E. (1976). "Responses to tones and noise of single cells in dorsal cochlear nucleus of unanesthetized cats." J Neurophys. 39: 282-300.